# The Distribution Family of Similarity Distances

**Gertjan J. Burghouts**[*]        **Arnold W. M. Smeulders**        **Jan-Mark Geusebroek** [†]

Intelligent Systems Lab Amsterdam
Informatics Institute
University of Amsterdam

## Abstract

Assessing similarity between features is a key step in object recognition and scene categorization tasks. We argue that knowledge on the distribution of distances generated by similarity functions is crucial in deciding whether features are similar or not. Intuitively one would expect that similarities between features could arise from any distribution. In this paper, we will derive the contrary, and report the theoretical result that $L_p$-norms –a class of commonly applied distance metrics– from one feature vector to other vectors are Weibull-distributed if the feature values are correlated and non-identically distributed. Besides these assumptions being realistic for images, we experimentally show them to hold for various popular feature extraction algorithms, for a diverse range of images. This fundamental insight opens new directions in the assessment of feature similarity, with projected improvements in object and scene recognition algorithms.

## 1   Introduction

Measurement of similarity is a critical asset of state of the art in computer vision. In all three major streams of current research - the recognition of known objects [13], assigning an object to a class [8, 24], or assigning a scene to a type [6, 25] - the problem is transposed into the equality of features derived from similarity functions. Hence, besides the issue of feature distinctiveness, comparing two images heavily relies on such similarity functions. We argue that knowledge on the *distribution* of distances generated by such similarity functions is even more important, as it is that knowledge which is crucial in deciding whether features are similar or not.

For example, Nowak and Jurie [21] establish whether one can draw conclusions on two never seen objects based on the similarity distances from known objects. Where they build and traverse a randomized tree to establish region correspondence, one could alternatively use the distribution of similarity distances to establish whether features come from the mode or the tail of the distribution. Although this indeed only hints at an algorithm, it is likely that knowledge of the distance distribution will considerably improve or speed-up such tasks.

As a second example, consider the clustering of features based on their distances. Better clustering algorithms significantly boost performance for object and scene categorization [12]. Knowledge on the distribution of distances aids in the construction of good clustering algorithms. Using this knowledge allows for the exact distribution shape to be used to determine cluster size and centroid, where now the Gaussian is often groundlessly assumed. We will show that in general distance distributions will strongly deviate from the Gaussian probability distribution.

A third example is from object and scene recognition. Usually this is done by measuring invariant feature sets [9, 13, 24] at a predefined raster of regions in the image or at selected key points in the image [11, 13] as extensively evaluated [17]. Typically, an image contains a hundred regions or a

---

[*]Dr. Burghouts is now with TNO Defense, The Netherlands, `gertjan.burghouts@tno.nl`.

[†]Corresponding author. Email: `mark@science.uva.nl`.

thousand key points. Then, the most expensive computational step is to compare these feature sets to the feature sets of the reference objects, object classes or scene types. Usually this is done by going over all entries in the image to all entries in the reference set and select the best matching pair. Knowledge of the distribution of similarity distances and having established its parameters enables a remarkable speed-up in the search for matching reference points and hence for matching images. When verifying that a given reference key-point or region is statistically unlikely to occur in this image, one can move on to search in the next image. Furthermore, this knowledge can well be applied in the construction of fast search trees, see e.g. [16].

Hence, apart from obtaining theoretical insights in the general distribution of similarities, the results derived in this paper are directly applicable in object and scene recognition.

Intuitively one would expect that the set of all similarity values to a key-point or region in an image would assume any distribution. One would expect that there is no preferred probability density distribution at stake in measuring the similarities to points or regions in one image. In this paper, we will derive the contrary. We will prove that under broad conditions the similarity values to a given reference point or region adhere to a class of distributions known as the Weibull distribution family. The density function has only three parameters: mean, standard deviation and skewness. We will verify experimentally that the conditions under which this result from mathematical statistics holds are present in common sets of images. It appears the theory predicts the resulting density functions accurately.

Our work on density distributions of similarity values compares to the work by Pekalska and Duin [23] assuming a Gaussian distribution for similarities. It is based on an original combination of two facts from statistical physics. An old fact regards the statistics of extreme values [10], as generated when considering the minima and maxima of many measurements. The major result of the field of extreme value statistics is that the probability density in this case can only be one out of three different types, independent of the underlying data or process. The second fact is a new result, which links these extreme value statistics to sums of correlated variables [2, 3]. We exploit these two facts in order to derive the distribution family of similarity measures.

This paper is structured as follows. In Section 2, we overview literature on similarity distances and distance distributions. In Section 3, we discuss the theory of distributions of similarity distances from one to other feature vectors. In Section 4, we validate the resulting distribution experimentally for image feature vectors. Finally, conclusions are given in Section 5.

## 2 Related work

### 2.1 Similarity distance measures

To measure the similarity between two feature vectors, many distance measures have been proposed [15]. A common metric class of measures is the $L_p$-norm [1]. The distance from one reference feature vector $s$ to one other feature vector $t$ can be formalized as:

$$d(s,t) = (\sum_{i=1}^{n} |s_i - t_i|^p)^{1/p}, \qquad (1)$$

where $n$ and $i$ are the dimensionality and indices of the vectors. Let the random variable $D_p$ represent distances $d(s,t)$ where $t$ is drawn from the random variable $T$ representing feature vectors. Independent of the reference feature vector $s$, the probability density function of $L_p$-distances will be denoted by $f(D_p = d)$.

### 2.2 Distance distributions

Ferencz *et al.* [7] have considered the Gamma distribution to model the $L_2$-distances from image regions to one reference region: $f(D_2 = d) = \frac{1}{\beta^\gamma \Gamma(\gamma)} d^{\gamma-1} e^{-d/\beta}$, where $\gamma$ is the shape parameter, and $\beta$ the scale parameter; $\Gamma(\cdot)$ denotes the Gamma function. In [7], the distance function was fitted efficiently from few examples of image regions. Although the distribution fits were shown to represent the region distances to some extent, the method lacks a theoretical motivation.

Based on the central limit theorem, Pekalska and Duin [23] assumed that $L_p$-distances between feature vectors are normally distributed: $f(D_p = d) = \frac{1}{\sqrt{2\pi}\beta} e^{-(d^2/\beta^2)/2}$. As the authors argue, the use of the central limit theorem is theoretically justified if the feature values are independent, identically distributed, and have limited variance. Although feature values generally have limited variance, unfortunately, they cannot be assumed to be independent and/or identically distributed as we will show below. Hence, an alternative derivation of the distance distribution function has to be followed.

## 2.3 Contribution of this paper

Our contribution is the theoretical derivation of a parameterized distribution for $L_p$-norm distances between feature vectors. In the experiments, we establish whether distances to image features adhere to this distribution indeed. We consider SIFT-based features [17], computed from various interest region types [18].

## 3 Statistics of distances between features

In this section, we derive the distribution function family of $L_p$-distances from a reference feature vector to other feature vectors. We consider the notation as used in the previous section, with $t$ a feature vector drawn from the random variable $T$. For each vector $t$, we consider the value at index $i$, $t_i$, resulting in a random variable $T_i$. The value of the reference vector at index $i$, $s_i$, can be interpreted as a sample of the random variable $T_i$. The computation of distances from one to other vectors involves manipulations of the random variable $T_i$ resulting in a new random variable: $X_i = |s_i - T_i|^p$. Furthermore, the computation of the distances $D$ requires the summation of random variables, and a reparameterization: $D = (\sum_{i=1}^{I} X_i)^{1/p}$. In order to derive the distribution of $D$, we start with the statistics of the summation of random variables, before turning to the properties of $X_i$.

### 3.1 Statistics of sums

As a starting point to derive the $L_p$-distance distribution function, we consider a lemma from statistics about the sum of random variables.

**Lemma 1** *For non-identical and correlated random variables $X_i$, the sum $\sum_{i=1}^{N} X_i$, with finite $N$, is distributed according to the generalized extreme value distribution, i.e. the Gumbel, Frechet or Weibull distribution.*

For a proof, see [2, 3]. Note that the lemma is an extension of the central limit theorem to non-identically distributed random variables. And, indeed, the proof follows the path of the central limit theorem. Hence, the resulting distribution of sums is different from a normal distribution, and rather one of the Gumbel, Frechet or Weibull distributions instead. This lemma is important for our purposes, as later the feature values will turn out to be non-identical and correlated indeed. To confine the distribution function further, we also need the following lemma.

**Lemma 2** *If in the above lemma the random variable $X_i$ are upper-bounded, i.e. $X_i < max$, the sum of the variables is Weibull distributed (and not Gumbel nor Frechet):*

$$f(Y = y) = \frac{\gamma}{\beta}(\frac{y}{\beta})^{\gamma-1} e^{-(\frac{y}{\beta})^\gamma} \ , \tag{2}$$

*with $\gamma$ the shape parameter and $\beta$ the scale parameter.*

For a proof, see [10]. Figure 1 illustrates the Weibull distribution for various shape parameters $\gamma$. This lemma is equally important to our purpose, as later the feature values will turn out to be upper-bounded indeed.

The combination of Lemmas 1 and 2 yields the distribution of sums of non-identical, correlated and upper-bounded random variables, summarized in the following theorem.

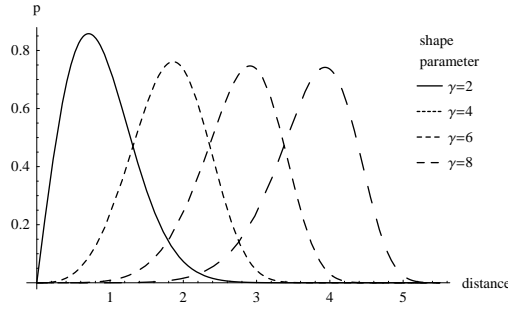

Figure 1: Examples of the Weibull distribution for various shape parameters $\gamma$.

**Theorem 1** *For non-identical, correlated and upper-bounded random variables $X_i$, the random variable $Y = \sum_{i=1}^{N} X_i$, with finite $N$, adheres to the Weibull distribution.*

The proof follows trivially from combining the different findings of statistics as laid down in Lemmas 1 and 2. Theorem 1 is the starting point to derive the distribution of $L_p$-norms from one reference vector to other feature vectors.

### 3.2 $L_p$-distances from one to other feature vectors

Theorem 1 states that $Y$ is Weibull-distributed, given that $\{X_i = |s_i - T_i|^p\}_{i\in[1,...,I]}$ are non-identical, correlated and upper-bounded random variables. We transform $Y$ such that it represents $L_p$-distances, achieved by the transformation $(\cdot)^{1/p}$:

$$Y^{1/p} = (\sum_{i=1}^{N} |s_i - T_i|^p)^{1/p}. \tag{3}$$

The consequence of the substitution $Z = Y^{1/p}$ for the distribution of $Y$ is a change of variables $z = y^{1/p}$ in Equation 2 [22]: $g(Z = z) = \frac{f(z^p)}{(1/p-1)z^{(1-p)}}$. This transformation yields a different distribution still of the Weibull type:

$$g(Z = z) = \frac{1}{(1/p - 1)} \frac{\gamma}{\beta^{1/p}} (\frac{z}{\beta^{1/p}})^{p\gamma-1} e^{-(\frac{z}{\beta^{1/p}})^{p\gamma}}, \tag{4}$$

where $\gamma' = p\gamma$ is the new shape parameter and $\beta' = \beta^{1/p}$ is the new scale parameter, respectively. Thus, also $Y^{1/p}$ and hence $L_p$-distances are Weibull-distributed under the assumed case.

We argue that the random variables $X_i = |s_i - T_i|^p$ and $X_j$ ($i \neq j$) are indeed non-identical, correlated and upper-bounded random variables when considering a set of values extracted from feature vectors at indices $i$ and $j$:

- $X_i$ and $X_j$ are upper-bounded. Features are usually an abstraction of a particular type of finite measurements, resulting in a finite feature. Hence, for general feature vectors, the values at index $i$, $T_i$, are finite. And, with finite $p$, it follows trivially that $X_i$ is finite.

- $X_i$ and $X_j$ are correlated. The experimental verification of this assumption is postponed to Section 4.1.

- $X_i$ and $X_j$ are non-identically distributed. The experimental verification of this assumption is postponed to Section 4.1.

We have obtained the following result.

**Corollary 1** *For finite-length feature vectors with non-identical, correlated and upper-bounded values, $L_p$ distances, for limited $p$, from one reference feature vector to other feature vectors adhere to the Weibull distribution.*

### 3.3 Extending the class of features

We extend the class of features for which the distances are Weibull-distributed. From now on, we allow the possibility that the vectors are preprocessed by a PCA transformation. We denote the PCA transform $g(\cdot)$ applied to a single feature vector as $s' = g(s)$. For the random variable $T_i$, we obtain $T_i'$. We are still dealing with upper-bounded variables $X_i' = |s_i' - T_i'|^p$ as PCA is a finite transform. The experimental verification of the assumption that PCA-transformed feature values $T_i'$ and $T_j'$, $i \neq j$ are non-identically distributed is postponed to Section 4.1. Our point here, is that we have assumed originally correlating feature values, but after the decorrelating PCA transform we are no longer dealing with correlated feature values $T_i'$ and $T_j'$. In Section 4.1, we will verify experimentally whether $X_i'$ and $X_j'$ correlate. The following observation is hypothesized. PCA translates the data to the origin, before applying an affine transformation that yields data distributed along orthogonal axes. The tuples $(X_i', X_j')$ will be in the first quadrant due to the absolute value transformation. Obviously, variances $\sigma(X_i')$ and $\sigma(X_j')$ are limited and means $\mu(X_i') > 0$ and $\mu(X_j') > 0$. For data constrained to the first quadrant and distributed along orthogonal axes, a negative covariance is expected to be observed. Under the assumed case, we have obtained the following result.

**Corollary 2** *For finite-length feature vectors with non-identical, correlated and upper-bounded values, and for PCA-transformations thereof, $L_p$ distances, for limited $p$, from one to other features adhere to the Weibull distribution.*

### 3.4 Heterogeneous feature vector data

We extend the corollary to hold also for composite datasets of feature vectors. Consider the composite dataset modelled by random variables $\{T_t\}$, where each random variable $T_t$ represents non-identical and correlated feature values. Hence, from Corollary 2 it follows that feature vectors from each of the $T_t$ can be fitted by a Weibull function $f^{\beta,\gamma}(d)$. However, the distances to each of the $T_t$ may have a different range and modus, as we will verify by experimentation in Section 4.1. For heterogeneous distance data $\{T_t\}$, we obtain a mixture of Weibull functions [14].

**Corollary 3 (Distance distribution)** *For feature vectors that are drawn from a mixture of datasets, of which each results in non-identical and correlated feature values, finite-length feature vectors with non-identical, correlated and upper-bounded values, and for PCA-transformations thereof, $L_p$ distances, for limited $p$, from one reference feature vector to other feature vectors adhere to the Weibull mixture distribution: $f(D = d) = \sum_{i=1}^{c} \rho_i \cdot f_i^{\beta_i,\gamma_i}(d)$, where $f_i$ are the Weibull functions and $\rho_i$ are their respective weights such that $\sum_{i=1}^{c} \rho_i = 1$.*

## 4 Experiments

In our experiments, we validate assumptions and Weibull goodness-of-fit for the region-based SIFT, GLOH, SPIN, and PCA-SIFT features on COREL data [5]. We include these features for two reasons as: a) they are performing well for realistic computer vision tasks and b) they provide different mechanisms to describe an image region [17]. The region features are computed from regions detected by the Harris- and Hessian-affine regions, maximally stable regions (MSER), and intensity extrema-based regions (IBR) [18]. Also, we consider PCA-transformed versions for each of the detector/feature combinations. For reason of its extensive use, the experimentation is based on the $L_2$-distance. We consider distances from 1 randomly drawn reference vector to 100 other randomly drawn feature vectors, which we repeat 1,000 times for generalization. In all experiments, the features are taken from multiple images, except for the illustration in Section 4.1.2 to show typical distributions of distances between features taken from single images.

### 4.1 Validation of the corollary assumptions for image features

#### 4.1.1 Intrinsic feature assumptions

Corollary 2 rests on a few explicit assumptions. Here we will verify whether the assumptions occur in practice.

**Differences between feature values are correlated.** We consider a set of feature vectors $T_j$ and the differences at index $i$ to a reference vector $s$: $X_i = |s_i - T_{ji}|^p$. We determine the significance of Pearson's correlation [4] between the difference values $X_i$ and $X_j$, $i \neq j$. We establish the percentage of significantly correlating differences at a confidence level of 0.05. We report for each feature the average percentage of difference values that correlate significantly with difference values at an other feature vector index.

As expected, the feature value differences correlate. For SIFT, 99% of the difference values are significantly correlated. For SPIN and GLOH, we obtain 98% and 96%, respectively. Also PCA-SIFT contains significantly correlating difference values: 95%. Although the feature's name hints at uncorrelated values, it does not achieve a decorrelation of the values in practice. For each of the features, a low standard deviation $< 5\%$ is found. This expresses the low variation of correlations across the random samplings and across the various region types.

We repeat the experiment for PCA-transformed feature values. Although the resulting values are uncorrelated by construction, their differences are significantly correlated. For SIFT, SPIN, GLOH, and PCA-SIFT, the percentages of significantly correlating difference values are: 94%, 86%, 95%, and 75%, respectively.

**Differences between feature values are non-identically distributed.** We repeat the same procedure as above, but instead of measuring the significance of correlation, we establish the percentage of significantly differently distributed difference values $X_i$ by the Wilcoxon rank sum test [4] at a confidence level of 0.05. For SIFT, SPIN, GLOH, and PCA-SIFT, the percentages of significantly differently distributed difference values are: 99%, 98%, 92%, and 87%. For the PCA-transformed versions of SIFT, SPIN, GLOH, and PCA-SIFT, we find: 62%, 40%, 64%, and 51%, respectively. Note that in all cases, correlation is sufficient to fulfill the assumptions of Corollary 2. We have illustrated that feature value differences are significantly correlated and significantly non-identically distributed. We conclude that the assumptions of Corollary 2 about properties of feature vectors are realistic in practice, and that Weibull functions are expected to fit distance distributions well.

### 4.1.2 Inter-feature assumptions

In Corollary 3, we have assumed that distances from one to other feature vectors are described well by a mixture of Weibulls, if the features are taken from different clusters in the data. Here, we illustrate that clusters of feature vectors, and clusters of distances, occur in practice. Figure 2a shows Harris-affine regions from a natural scene which are described by the SIFT feature. The distances are described well by a single Weibull distribution. The same hold for distances from one to other regions computed from a man-made object, see Figure 2b. In Figure 2c, we illustrate the distances of one to other regions computed from a composite image containing two types of regions. This results in two modalities of feature vectors hence of similarity distances. The distance distribution is therefore bimodal, illustrating the general case of multimodality to be expected in realistic, heterogeneous image data. We conclude that the assumptions of Corollary 3 are realistic in practice, and that the Weibull function, or a mixture, fits distance distributions well.

### 4.2 Validation of Weibull-shaped distance distributions

In this experiment, we validate the fitting of Weibull distributions of distances from one reference feature vector to other vectors. We consider the same data as before. Over 1,000 repetitions we consider the goodness-of-fit of $L_2$-distances by the Weibull distribution. The parameters of the Weibull distribution function are obtained by maximum likelihood estimation. The established fit is assessed by the Anderson-Darling test at a confidence level of $\alpha = 0.05$ [20]. The Anderson-Darling test has also proven to be suited to measure the goodness-of-fit of mixture distributions [19].

Table 1 indicates that for most of the feature types computed from various regions, more than 90% of the distance distributions is fit by a single Weibull function. As expected, distances between each of the SPIN, SIFT, PCA-SIFT and GLOH features, are fitted well by Weibull distributions. The exception here is the low number of fits for the SIFT and SPIN features computed from Hessian-affine regions. The distributions of distances between these two region/feature combinations tend to have multiple modes. Likewise, there is a low percentage of fits of $L_2$-distance distributions of the

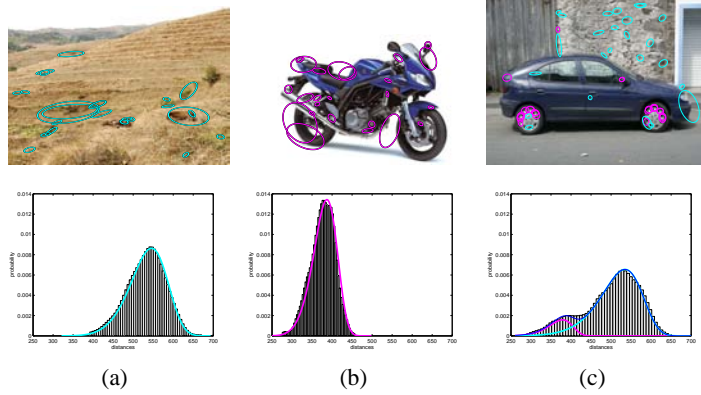

|  | (a) | (b) | (c) |

Figure 2: Distance distributions from one randomly selected image region to other regions, each described by the SIFT feature. The distance distribution is described by a single Weibull function for a natural scene (a) and a man-made object (b). For a composite image, the distance distribution is bimodal (c). Samples from each of the distributions are shown in the upper images.

Table 1: Accepted Weibull fits for COREL data [5].

|  | Harris-affine | | Hessian-affine | | MSER | | IBR | |
|---|---|---|---|---|---|---|---|---|
|  | $c = 1$ | $c \leq 2$ | $c = 1$ | $c \leq 2$ | $c = 1$ | $c \leq 2$ | $c = 1$ | $c \leq 2$ |
| SIFT | 95% | 100% | 60% | 99% | 98% | 100% | 92% | 100% |
| SIFT ($g$ =PCA) | 95% | 99% | 60% | 98% | 98% | 100% | 92% | 99% |
| PCA-SIFT | 89% | 100% | 96% | 100% | 94% | 100% | 95% | 100% |
| PCA-SIFT ($g$ =PCA) | 89% | 100% | 96% | 100% | 94% | 100% | 95% | 100% |
| SPIN | 71% | 99% | 12% | 99% | 77% | 99% | 45% | 98% |
| SPIN ($g$ =PCA) | 71% | 100% | 12% | 97% | 77% | 99% | 45% | 98% |
| GLOH | 87% | 100% | 91% | 100% | 82% | 99% | 86% | 100% |
| GLOH ($g$ =PCA) | 87% | 100% | 91% | 99% | 82% | 99% | 86% | 100% |

*Percentages of $L_2$-distance distributions fitted by a Weibull function ($c = 1$) and a mixture of two Weibull functions ($c \leq 2$) are given.*

SPIN feature computed from IBR regions. Again, multiple modes in the distributions are observed. For these distributions, a mixture of two Weibull functions provides a good fit ($\geq 97\%$).

## 5   Conclusion

In this paper, we have derived that similarity distances between one and other image features in databases are Weibull distributed. Indeed, for various types of features, i.e. the SPIN, SIFT, GLOH and PCA-SIFT features, and for a large variety of images from the COREL image collection, we have demonstrated that the similarity distances from one to other features, computed from $L_p$ norms, are Weibull-distributed. These results are established by the experiments presented in Table 1. Also, between PCA-transformed feature vectors, the distances are Weibull-distributed. The Malahanobis distance is very similar to the $L_2$-norm computed in the PCA-transformed feature space. Hence, we expect Mahalanobis distances to be Weibull distributed as well. Furthermore, when the dataset is a composition, a mixture of few (typically two) Weibull functions suffices, as established by the experiments presented in Table 1. The resulting Weibull distributions are distinctively different from the distributions suggested in literature, as they are positively or negatively skewed while the Gamma [7] and normal [23] distributions are positively and non-skewed, respectively.

We have demonstrated that the Weibull distribution is the preferred choice for estimating properties of similarity distances. The assumptions under which the theory is valid are realistic for images. We experimentally have shown them to hold for various popular feature extraction algorithms, and for a diverse range of images. This fundamental insight opens new directions in the assessment of feature similarity, with projected improvements and speed-ups in object/scene recognition algorithms.

**Acknowledgments**

This work is partly sponsored by the EU funded NEST project PERCEPT, by the Dutch BSIK project Multimedian, and by the EU Network of Excellence MUSCLE.

## References

[1] B. G. Batchelor. *Pattern Recognition: Ideas in Practice*. Plenum Press, New York, 1995.

[2] E. Bertin. Global fluctuations and Gumbel statistics. *Physical Review Letters*, 95(170601):1–4, 2005.

[3] E. Bertin and M. Clusel. Generalised extreme value statistics and sum of correlated variables. *Journal of Physics A*, 39:7607, 2006.

[4] W. J. Conover. *Practical nonparametric statistics*. Wiley, New York, 1971.

[5] Corel Gallery. www.corel.com.

[6] L. Fei-Fei and P. Perona. A bayesian hierarchical model for learning natural scene categories. In *CVPR*, 2005.

[7] A. Ferencz, E.G. Learned-Miller, and J. Malik. Building a classification cascade for visual identification from one example. In *Proceedings of the International Conference Computer Vision*, pages 286–293. IEEE Computer Society, 2003.

[8] R. Fergus, P. Perona, and A. Zisserman. A sparse object category model for efficient learning and exhaustive recognition. In *Proceedings of the Computer Vision and Pattern Recognition*. IEEE, 2005.

[9] J. M. Geusebroek, R. van den Boomgaard, A. W. M. Smeulders, and H. Geerts. Color invariance. *IEEE Transactions on Pattern Analysis and Machine Intelligence*, 23(12):1338–1350, 2001.

[10] E. J. Gumbel. *Statistics of Extremes*. Columbia University Press, New York, 1958.

[11] C. Harris and M. Stephans. A combined corner and edge detector. In *Proceedings of the 4th Alvey Vision Conference*, pages 189–192, Manchester, 1988.

[12] F. Jurie and B. Triggs. Creating efficient codebooks for visual recognition. In *ICCV*, pages 604–610, 2005.

[13] D. G. Lowe. Distinctive image features from scale-invariant keypoints. *International Journal of Computer Vision*, 60(2):91–110, 2004.

[14] J. M. Marin, M. T. Rodriquez-Bernal, and M. P. Wiper. Using weibull mixture distributions to model heterogeneous survival data. *Communications in statistics*, 34(3):673–684, 2005.

[15] R. S. Michalski, R. E. Stepp, and E. Diday. A recent advance in data analysis: Clustering objects into classes characterized by conjunctive concepts. In L. N. Kanal and A. Rosenfeld, editors, *Progress in Pattern Recognition*, pages 33–56. North-Holland Publishing Co., Amsterdam, 1981.

[16] K. Mikolajczyk, B. Leibe, and B. Schiele. Multiple object class detection with a generative model. In *CVPR*, 2006.

[17] K. Mikolajczyk and C. Schmid. A performance evaluation of local descriptors. *IEEE Transactions on Pattern Analysis and Machine Intelligence*, 27(10):1615–1630, 2005.

[18] K. Mikolajczyk, T. Tuytelaars, C. Schmid, A. Zisserman, J. Matas, F. Schaffalitzky, T. Kadir, and L. Van Gool. A comparison of affine region detectors. *International Journal of Computer Vision*, 65(1/2):43–72, 2005.

[19] K. Mosler. Mixture models in econometric duration analysis. *Applied Stochastic Models in Business and Industry*, 19(2):91–104, 2003.

[20] NIST/SEMATECH. *e-Handbook of Statistical Methods*. NIST, http://www.itl.nist.gov/div898/handbook/, 2006.

[21] E. Nowak and F. Jurie. Learning visual similarity measures for comparing never seen objects. In *CVPR*, 2007.

[22] A. Papoulis and S. U. Pillai. *Probability, Random Variables and Stochastic Processes*. McGraw-Hill, New York, 4 edition, 2002.

[23] E. Pekalska and R. P. W. Duin. Classifiers for dissimilarity-based pattern recognition. In *Proceedings of the International Conference on Pattern Recognition*, volume 2, page 2012, 2000.

[24] C. Schmid and R. Mohr. Local grayvalue invariants for image retrieval. *IEEE Transactions on Pattern Analysis and Machine Intelligence*, 19(5):530–535, 1997.

[25] J.C. van Gemert, J.M. Geusebroek, C.J. Veenman, C.G.M. Snoek, and Arnold W.M. Smeulders. Robust scene categorization by learning image statistics in context. In *CVPR Workshop on Semantic Learning Applications in Multimedia (SLAM)*, 2006.

